# GPPS: A Gaussian Process Positioning System for Cellular Networks

**Anton Schwaighofer**[*] **Marian Grigoraş, Volker Tresp, Clemens Hoffmann**
Siemens Corporate Technology, Information and Communications
81730 Munich, Germany
`http://www.igi.tugraz.at/aschwaig`

## Abstract

In this article, we present a novel approach to solving the localization problem in cellular networks. The goal is to estimate a mobile user's position, based on measurements of the signal strengths received from network base stations. Our solution works by building Gaussian process models for the distribution of signal strengths, as obtained in a series of calibration measurements. In the localization stage, the user's position can be estimated by maximizing the likelihood of received signal strengths with respect to the position. We investigate the accuracy of the proposed approach on data obtained within a large indoor cellular network.

## 1 Introduction

Cellular networks form the basis of modern wireless communication infrastructure. Examples include GSM and UMTS networks for mobile phones, wireless LAN (WLAN) for computer networks, and DECT for cordless phones. Within these networks, location-based services (services that are tailored specifically to the current position of the mobile user) have great potential. Examples of such services are guiding the user through a building or city, delivering the time-table of buses at the nearest bus stop, or simply answering the user's query "Where am I?". All such services crucially depend on methods to accurately estimate the position of the mobile user within the network ("localization", "positioning").

In this article, we present a novel approach to obtain position estimates for the mobile user. Most importantly, this method is based solely on infrastructure that is already present in a typical cellular network, and thus leads to minimal extra cost. Furthermore, we focus on indoor networks, where a number of specific problems needs to be addressed. Since our approach relies heavily on Gaussian process models, we call it the "Gaussian process positioning system" (GPPS).

We proceed by introducing the localization problem in detail in Sec. 1.1, and by giving a brief overview of previous approaches. Sec. 2 follows with a description of the Gaussian process positioning system (GPPS). Sec. 3 shows how the required calibration stage of the system can be performed in an optimal manner. Sec. 4 presents an evaluation of the

---

[*]Also with the Institute for Theoretical Computer Science, Graz University of Technology, Austria

GPPS in a DECT network environment. We show that the GPPS gives accurate location estimates, in particular when only very few calibration measurements are available.

## 1.1 Problem Description

Our overall goal is to develop a localization system for indoor cellular networks, that is (in order to minimize cost) based solely on existing standard networking hardware. Location estimates can be based on different characteristics of the radio signal received at the mobile station (i.e., the laptop in a WLAN network, or the phone in a DECT network). Yet, in most hardware, the only available information about the radio signal is the received signal strength. Information like phase or propagation time from the base station requires additional hardware, and can thus not be used.

In general, estimating the user's position based only on measurements of the signal strength is known to be a very challenging task [7], in particular in indoor networks. Due to reflections, refraction, and scattering of the electromagnetic waves along structures of the building, the received signal is only a distorted version of the transmitted signal. Noise and co-channel interference further corrupt the signals [4]. Furthermore, when using standard hardware, we must expect a high level of measurement noise for the signal strength. Changes in the environment can also have a strong impact on signal propagation. For example, in a WLAN environment [1], it has been noted that shielding by a single person can attenuate the signal by up to $-3.5$ dBm. Also, the localization system ought to be robust, since base stations may fail, be switched off, or may be temporarily shielded for unknown reasons. In these cases, a sensible localization system should not draw the conclusion that the user is far from the respective base station.

Due to the complex signal propagation behaviour, almost all previous approaches to indoor localization use an initial calibration stage. Calibration here means that signal strengths received from the network base stations are measured at a number of points inside the building. Systems differ in their ways of using this calibration data. In principle, two basic approaches can be used here. In a "forward modelling" approach, a model of signal strength as a function of position is built first. The localization procedure then tries to find the location which best agrees with the measured signal strengths. Alternatively, the mapping from signal strengths to position can be modelled directly ("inverse modelling").

The RADAR system [1], one of the first indoor localization systems, is an inverse modelling approach using a nearest neighbor technique. [7] build simple probabilistic models from the calibration data (forward modelling), in conjunction with maximum likelihood position estimation. Bayesian networks have been considered by [2], with states of node corresponding to different locations (using coarse discretization). Discrete locations, yet with a finer grid, are also considered in [5], in an approach inspired by robot navigation.

## 2 The Gaussian Process Positioning System

The difficulties of indoor localization, as mentioned in Sec. 1.1, call for a probabilistic method for localization. The key idea of the Gaussian process positioning system (GPPS) is to use Gaussian process models for the signal strength received from each base station, and to obtain position estimates via maximum likelihood, i.e. by searching for the position which best fits the measured signal strengths.

Consider a cellular network with a total of $B$ base stations. Assume that, for each of base stations, we have a probabilistic model that describes the distribution of received signal strength. More formally, we denote by $p_j(s_j \,|\, \mathbf{t})$ the likelihood of receiving a signal strength $s_j$ from the $j$-th base station on position $\mathbf{t}$.

With the models $p_j(s_j|\mathbf{t})$, $j = 1,\ldots,B$ given, localization can be done in a straight-forward way. The user reports a vector $\mathbf{s}$ (of length $B$) of signal strength measurements for all base stations. It may occur that no signal is received from some base stations (indicated by $s_j = \emptyset$), e.g., because the user is too far from this base station, or due to hardware failure. In the GPPS, the estimated position $\hat{\mathbf{t}}$ is computed by maximizing the joint likelihood[1] with respect to the unknown position,

$$\hat{\mathbf{t}} = \arg\max_{\mathbf{t}} \prod_{j:s_j \neq \emptyset} p_j(s_j|\mathbf{t}). \tag{1}$$

In the above equation, we only use the likelihood contributions of those base stations that are actually received. Alternatively, one could use a very low signal strength as a default value for each base station that is not received [7]. We found that this can give high errors if a base station close to the user fails, since now the low default value indicates that one should expect the user to be far from the base station. Thus, by using the above expression, we also obtain a certain degree of robustness with respect to hardware failures or other unexpected effects.

Yet, we still need to define and build suitable base station models $p_j(s_j|\mathbf{t})$, $j = 1,\ldots,B$. In the GPPS, we use Gaussian process (GP) models for this task, where each base station model is estimated from the calibration data. Gaussian processes are particularly useful here for several reasons. Firstly, one obtains a full predictive distribution, as opposed to the point estimate output by other regression approaches. Secondly, GPs are a nonparametric method that can flexibly adapt to the complex signal propagation behaviour observed in indoor cellular networks.

Mind that this approach opens a wide range of possibilities for further extensions. Due to particular project requirements, we currently only use the maximum likelihood position estimate in Eq. (1) ("one-shot localization" without error estimates). Instead of the implicitly assumed uninformative prior in Eq. (1), one could, for example, specify an informative prior based on known previous positions of the user, in conjunction with a motion model. Subsequently, the complete posterior distribution $p(\mathbf{t}|\mathbf{s})$ can be evaluated for localization.

In the following sections, we will describe the GP models in more detail, and also discuss the choice of kernel function, which is of great importance in order to build an accurate localization system.

## 2.1 Gaussian Process Models for Signal Strengths

In the GPPS, a Gaussian process (GP) approach is used for the models $p_j(s_j|\mathbf{t})$ that describe the signal strength received from a single base station $j$. Details on GP models can be found, for example, in [6]; we only give a brief summary here.

Recall from Sec. 1.1 that the proposed GPPS is based on a set calibration measurements, where the signal strength is measured at a number of points spread over the area to be covered. Consider now the calibration data for a single base station $j$. We denote this calibration data by $\mathcal{D}_j = \{(\mathbf{x}_i, y_i)\}_{i=1}^N$, meaning that a signal strength of $y_i$ has been measured on point $\mathbf{x}_i$, with a total of $N$ calibration measurements.

For simplicity of computation, we use a GP model with Gaussian noise, i.e., the measured signal strength $y_i$ is composed of a "true" signal strength $s(\mathbf{x}_i)$ plus independent Gaussian (measurement) noise $e_i$ of variance $\sigma^2$, with $y_i = s(\mathbf{x}_i) + e_i$. The Gaussian process assumption for the true signal $s$ implies that the true signal strengths for all calibration

points $(s(\mathbf{x}_1), \ldots, s(\mathbf{x}_N))$ are jointly Gaussian distributed, with zero mean and covariance matrix $K$. $K$ itself is given by the kernel (covariance) function $k$, with $K_{mn} = k(\mathbf{x}_m, \mathbf{x}_n)$, $m, n = 1, \ldots, N$.

Given the calibration data $\mathcal{D}_j$, the predictive distribution for the signal strength $s_j$ received on some arbitrary point $\mathbf{t}$ turns out to be Gaussian. With $\mathbf{v}(\mathbf{t}) = (k(\mathbf{t}, \mathbf{x}_1), \ldots, k(\mathbf{t}, \mathbf{x}_N))^\top$, $\mathbf{y} = (y_1, \ldots, y_N)^\top$ and $Q = K + \sigma^2 I$, mean and variance of the prediction are

$$\mathrm{E}(s_j \,|\, \mathcal{D}_j, \mathbf{t}) = \mathbf{v}(\mathbf{t})^\top Q^{-1} \mathbf{y} \tag{2}$$

$$\mathrm{var}(s_j \,|\, \mathcal{D}_j, \mathbf{t}) = k(\mathbf{t}, \mathbf{t}) - \mathbf{v}(\mathbf{t})^\top Q^{-1} \mathbf{v}(\mathbf{t}) \tag{3}$$

Using these expressions for the predictive distribution (a univariate Gaussian) in Eq. (1) becomes straight forward. Also, gradients of the likelihood with respect to the position $\mathbf{t}$ can be derived easily [8]. Thus, the position estimate, Eq. (1), can be computed easily using either some standard optimization routine, or by evaluating the likelihood grid-based in the area of interest.

An important issue is also the choice of noise variance $\sigma^2$ and the parameters $\theta$ of the kernel function $k$ (which we have not explicitly denoted above) . We set them by maximizing the marginal likelihood of the calibration data with respect to the model parameters, which turns out to be [6]

$$(\hat{\sigma}^2, \hat{\theta}) = \arg\max_{\sigma^2, \theta} \left( -\log \det Q - \mathbf{y}^\top Q^{-1} \mathbf{y} \right). \tag{4}$$

The model parameters $(\hat{\sigma}^2, \hat{\theta})$ are set individually for each base station.

## 2.2 The Matérn Kernel Function

In our GPPS application, with a 2-dimensional input space for the GP models, the choice of an appropriate kernel function is a more critical issue if compared to typical machine learning applications with many input dimensions. For the commonly used squared exponential kernel, $k(\mathbf{x}, \mathbf{x}') = \exp(-w\|\mathbf{x} - \mathbf{x}'\|^2)$, it has been argued [9] that sample paths of such GP models are "infinitely smooth", thus often leading to unreasonably low predictive variance. In GPPS, we instead use the Matérn class of kernel functions [9], which allows a continuous parameterization of the smoothness of the sample paths via its parameter $\nu$. Its functional form is

$$k(\mathbf{x}, \mathbf{x}') = M_\nu(z) = \frac{2\left(\sqrt{\nu}z\right)^\nu}{\Gamma(\nu)} K_\nu(2\sqrt{\nu}z) \tag{5}$$

where $\Gamma(\nu)$ is the Gamma function and $K_\nu(r)$ is the modified Bessel function of the second kind of degree $\nu$. The parameter $\nu$ determines the smoothness (fractal dimension) of the sample paths and can be estimated from the data using Eq. (4). We use an isotropic kernel function with length scale $w$, thus $z^2 = w\|\mathbf{x} - \mathbf{x}'\|^2$.

## 2.3 Learning GP Models with Matérn Kernel

For efficient solutions of Eq. (4), we require derivatives of the Matérn kernel function Eq. (5) with respect to all its parameters $\nu, w$. Numerical gradients, as used for example by [9], require a large number of evaluations of the Bessel functions and thus lead to a huge computational overhead. To compute the derivatives analytically, we use

$$\frac{\partial \Gamma(\nu)}{\partial \nu} = \Gamma(\nu)\Psi(\nu) \quad \text{and} \quad \frac{\partial K_\nu(z)}{\partial z} = -\frac{1}{2}\left(K_{\nu-1}(z) + K_{\nu+1}(z)\right) \tag{6}$$

where $\Psi(\nu)$ is the Polygamma function of order 0. To the best of our knowledge, there is no closed form expression for the derivative of the Bessel function $K_\nu(z)$ with respect to its

degree $\nu$. We approximate this by $\frac{\partial K_\nu(z)}{\partial \nu} = DK_\nu(z) \approx \varepsilon^{-1}(K_{\nu+\varepsilon}(z) - K_\nu(z))$. Using these identities, we find for the gradients of the Matérn function, Eq. (5),

$$\frac{\partial M_\nu(z)}{\partial z} = \frac{\nu}{z} M_\nu(z) - \frac{2\sqrt{\nu}\left(\sqrt{\nu z}\right)^\nu}{\Gamma(\nu)}\left(K_{\nu-1}(2\sqrt{\nu z}) + K_{\nu+1}(2\sqrt{\nu z})\right)$$

$$\frac{\partial M_\nu(z)}{\partial \nu} = M_\nu(z)\left(\frac{1}{2} + \log\left(\sqrt{\nu z}\right) - \Psi(\nu)\right) \tag{7}$$

$$+ \frac{2\left(\sqrt{\nu z}\right)^\nu}{\Gamma(\nu)}\left(-\frac{z}{2\sqrt{\nu}}\left(K_{\nu-1}(2\sqrt{\nu z}) + K_{\nu+1}(2\sqrt{\nu z})\right) + DK_\nu(2\sqrt{\nu z})\right).$$

Based on the above equations, derivatives of Eq. (4) with respect to the model parameters $\sigma^2, \nu, w$ can be computed using standard matrix algebra, see [6].

## 3  Optimal Calibration and Model Building

In order to make the GPPS, as presented in Sec. 2, a practical system, two further issues need to be solved. Firstly, it must be noted that taking calibration measurements is a very time-consuming (thus, expensive) task. The number of calibration data must thus be kept as low as possible, while retaining high localization accuracy. This question has been addressed in the literature under the name optimal design. [3] showed that—in a 2-dimensional space—hexagonal sampling design yields optimal results in terms of integral mean square error when the covariance structure of the underlying Gaussian process is unknown. We also adopt this optimal design for the GPPS system when evaluating it in Sec. 4.

Secondly, we assumed a GP model with zero mean in Sec. 2.1, which clearly does not fit the propagation law of radio signals. In the actual GPPS, the GP mean is a linear function of the distance to the base station (when signal strength is given on a logarithmic scale).

The overall process of building the GPPS is summarized is follows. Starting point is the calibration data, with a total of $C$ measurements. On calibration point $\mathbf{x}_i, i \in \{1, \ldots, C\}$, we receive a signal strength of $c_{ij}$ from base station $j$, $j \in \{1, \ldots, B\}$, or $c_{ij} = \emptyset$ if base station $j$ has not been received at $\mathbf{x}_i$ (for example, due to signal obstruction). Signal strength is measured in dB, all model fitting is thus done on a logarithmic scale.

The calibration data is then split into subsets $\mathcal{D}_j$ containing those points where base station $j$ has actually been received, i.e., $\mathcal{D}_j = \{(\mathbf{x}_i, c_{ij}) : c_{ij} \neq \emptyset\}$, corresponding to $\mathcal{D}_j$ introduced in Sec. 2.1. For each base station, that is, for each data $\mathcal{D}_j$, we proceed as follows:

1. Often, the exact position of base station $j$ is not known.[2] In this case, we use a simple estimate for the base station position, that is the average of the 3 calibration points $\mathbf{x}_i$ with maximum signal strength $y_i$. This estimate is rather crude, yet we found it to give sensible results in all of the configurations we have considered. In particular with sparse calibration measurements, more sophisticated estimates for the base station position are difficult to come up with.

2. Compute the distance of each calibration point to the base station (using either the exact or the estimated position obtained in step 1). As the mean function of the GP model, we fit a linear model[3] to the received signal strength as a function of distance to the base station. Subtract the value of the mean function from the

original measurements, and use the modified values in the subsequent GP model fitting procedure.

3. Use Eq. (4) to find optimal parameters for the GP model, which are the noise variance $\sigma^2$, the Matérn smoothness parameter $\nu$ and the input length scale $w$.

## 4 Evaluation in a DECT Network

We tested the accuracy of the GPPS in a large DECT cellular network. In a large assembly hall of $250 \times 180$ meters, measurements of signal strengths received from DECT base stations were made on 650 points spread over the hall. In this environment, moving robots, metal constructions, corridors, office cubicles, etc., are all affecting the signal propagation. We observed a very high fluctuation of received signals (up to $\pm 10$ dB when repeating measurements, while the total signal range is only $-30$ to $-90$ dB), both due to measurement noise, and due to dynamical changes of the environment.

We compare the GPPS with a nearest neighbor based localization system (abbreviated by NNLoc in the following), that is quite similar to the RADAR [1] approach.[4] This system finds the calibration measurements that best match the signal strength received at test stage. The best matches are used in a weighted triangulation scheme to compute the location estimate. This method requires careful fine tuning of parameters, and we have to omit details for brevity here.

**Dense Calibration Points**   In a first experiment, we investigate the achievable precision of location estimates when using the full set of calibration measurements. We evaluate both the GPPS and the nearest neighbor based method in a 5fold cross validation scheme. The total set of measurements is split up into five equally sized parts, where four of these parts were used as the calibration set. The resulting positioning system is tested on the fifth part left out. This is repeated five times, so that each point is being used as the test point exactly once. We found that, in this setting, the nearest neighbor based method NNLoc works very fine, and provides an average localization error of 7 meters. The GPPS performs slightly worse, with an average error of 7.5 meters. With the GPPS, localization is typically based on around 15 base stations, that is, 15 likelihood terms contributing to Eq. (1).

Unfortunately, such a high number of calibration measurements is unlikely to be available in practice. Taking calibration measurements is a very costly process, in particular if larger areas need to be covered. Thus, one is very much interested in keeping the number of calibration points as low as possible.

**Experiments with Sparse Calibration Points**   In the second experimental setup, we aim at building the positioning system with only a minimal number of calibration points. Again, 5fold cross validation was performed. After splitting the data into five parts, we select subsets of $\tilde{C} = 100, 50, 25, 12$ points, either at random or simulating the optimal design, from the union of four of these parts. The localization system is built based on these $\tilde{C}$ points and evaluated on the fifth part of the data. In order to simulate a near-optimal design (see Sec. 3), we superimpose a hexagonal grid with $\tilde{C}$ points on the area under consideration. Out of the given calibration measurements, we select those $\tilde{C}$ points that are closest (in terms of Euclidean distance) to the grid points.

In Fig. 1 we plot the localization accuracy, averaged over the 5fold cross validation, of the GPPS and the nearest neighbor based system built on only $\tilde{C}$ calibration points,

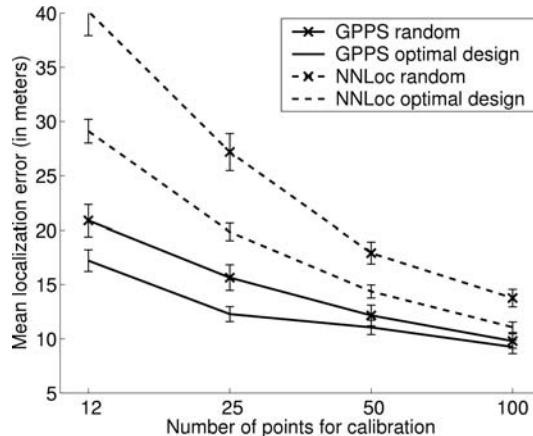

Figure 1: Mean localization error of the GPPS and the NNLoc method, as a function of the number of calibration points used. Vertical bars indicate $\pm 1$ standard deviation of the mean localization error. The calibration points are either selected at random, or according to an optimal design criterion

$\tilde{C} \in \{100, 50, 25, 12\}$ calibration measurement. It can be clearly seen that the GPPS system (with optimal design) achieves a high precision for its location estimates, even when using only a minimal number of calibration measurements. With only 12 calibration measurements, GPPS achieves an average error of around 17 meters, while the competing method reaches only 29 meters at best. In this setting, the average distance in between calibration measurements is around 75 meters. Both the NNLoc system and the GPPS system show large improvements of performance when selecting the calibration points according to the optimal design, instead of a purely random fashion. Also, note that the localization error of the GPPS system degrades only slowly when the number of calibration measurements is reduced. In contrast, the curves for the nearest neighbor based method show a sharper increase of positioning error.

It is worth noticing that the choice of kernel functions has a strong impact on the localization accuracy of the GPPS. In Fig. 2(a), we also plot a comparison of the GPPS with either the Matérn kernel, Eq. (5), or an RBF kernel of the form $k(\mathbf{x}, \mathbf{x}') = \exp(-w\|\mathbf{x} - \mathbf{x}'\|)$. GP models with RBF kernels tend to be over-optimistic [9] about the predictive variance, Eq. (3), which in turn leads to overly tight position estimates. Thus, the accuracy of GPPS with RBF kernel is clearly inferior to that of GPPS with Matérn kernel. It is also interesting to consider different methods for selecting the calibration points. Fig. 2(b) plots the accuracy obtained with GPPS, when calibration points are either placed randomly, on a hexagonal grid (the theoretically optimal procedure) or on a square grid. Somehow counter-intuitively, a square grid for calibration gives a performance that is just as good or even worse than a random grid. In contrast, localization with NNLoc performs about the same with either hexagonal or square grid (this is not plotted in the figure).

## 5   Conclusions

In this article, we presented a novel approach to solving the localization problem in indoor cellular network networks. Gaussian process (GP) models with the Matérn kernel function were used as models for individual base stations, so that location estimates could be computed using maximum likelihood. We showed that this new Gaussian process positioning system (GPPS) can provide sufficiently high accuracy when used within a DECT network.

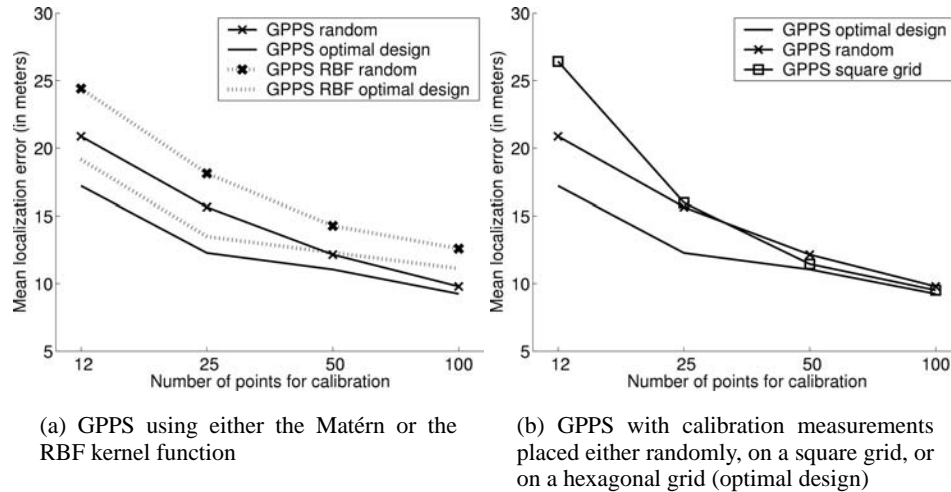

(a) GPPS using either the Matérn or the RBF kernel function

(b) GPPS with calibration measurements placed either randomly, on a square grid, or on a hexagonal grid (optimal design)

Figure 2: Average localization error of the GPPS method with different kernel function (left) and different methods for placing calibration points (right)

A particular advantage of the GPPS system is that it can be based on only a small number of calibration measurements, and yet retain high accuracy. Furthermore, we showed how calibration points can be optimally chosen in order to provide high accuracy position estimates.

**Acknowledgments** Anton Schwaighofer gratefully acknowledges support through an Ernst-von-Siemens scholarship.

## Footnotes

[1]Assuming independence of the individual measurements. One could also use a solution inspired from co-kriging, that takes into account the full dependence between signals received from different base stations. We did not consider this solution for reasons of efficiency.

[2]When setting up the network, or after modifying the network by moving base stations, the base station positions are often not recorded.

[3]Alternatively, one could also use a procedure similar to universal kriging, and combine fitting of the mean function with learning the parameters of the kernel function, see Eq. (4).

[4]We also investigated localization using Eq. (1) with a simplistic propagation model, where the expected signal (on log scale) is a linear function of the distance to the base station. Yet, this approach lead to very poor localization accuracy, and is thus not considered in more detail here.

## References

[1] Bahl, P., Padmanabhan, V. N., and Balachandran, A. A software system for locating mobile users: Design, evaluation, and lessons, 2000. Revised version of Microsoft Research Technical Report MSR-TR-2000-12, available from the authors webpages.

[2] Castro, P., Chiu, P., Kremenek, T., and Muntz, R. A probabilistic room location service for wireless network environments. In *Proceedings of the 3rd International Conference on Ubiquitous Computing (Ubicomp 2001)*. 2001.

[3] Hamprecht, F. A. and Agrell, E. Exploring a space of materials: Spatial sampling design and subset selection. In J. N. Cawse, ed., *Experimental Design for Combinatorial and High Throughput Materials Development*. John Wiley & Sons, 2002.

[4] Hashemi, H. The indoor radio propagation channel. *Proceedings of the IEEE*, 81(7):943–968, 1993.

[5] Ladd, A. M., Bekris, K. E., Rudys, A., Marceau, G., Kavraki, L. E., and Wallach, D. S. Robotics-based location sensing using wireless ethernet. In *Proceedings of the Eight ACM International Conference on Mobile Computing and Networking (MOBICOM 2002)*. 2002.

[6] Rasmussen, C. E. *Evaluation of Gaussian Processes and other methods for non-linear regression*. Ph.D. thesis, University of Toronto, 1996.

[7] Roos, T., Myllymäki, P., Tirri, H., Misikangas, P., and Sievänen, J. A probabilistic approach to WLAN user location estimation. *International Journal of Wireless Information Networks*, 9(3):155–164, 2002.

[8] Schwaighofer, A. *Kernel Systems for Regression and Graphical Modelling*. Ph.D. thesis, Institute for Theoretical Computer Science, Graz University of Technology, Austria, 2003.

[9] Stein, M. *Interpolation of Spatial Data. Some Theory for Kriging*. Springer Verlag, 1999.